# A tighter bound for graphical models

**M.A.R. Leisink\* and H.J. Kappen[†]**
Department of Biophysics
University of Nijmegen, Geert Grooteplein 21
NL 6525 EZ Nijmegen, The Netherlands
{martijn,bert}@mbfys.kun.nl

## Abstract

We present a method to bound the partition function of a Boltzmann machine neural network with any odd order polynomial. This is a direct extension of the mean field bound, which is first order. We show that the third order bound is strictly better than mean field. Additionally we show the rough outline how this bound is applicable to sigmoid belief networks. Numerical experiments indicate that an error reduction of a factor two is easily reached in the region where expansion based approximations are useful.

## 1 Introduction

Graphical models have the capability to model a large class of probability distributions. The neurons in these networks are the random variables, whereas the connections between them model the causal dependencies. Usually, some of the nodes have a direct relation with the random variables in the problem and are called 'visibles'. The other nodes, known as 'hiddens', are used to model more complex probability distributions.

Learning in graphical models can be done as long as the likelihood that the visibles correspond to a pattern in the data set, can be computed. In general the time it takes, scales exponentially with the number of hidden neurons. For such architectures one has no other choice than using an approximation for the likelihood.

A well known approximation technique from statistical mechanics, called Gibbs sampling, was applied to graphical models in [1]. More recently, the mean field approximation known from physics was derived for sigmoid belief networks [2]. For this type of graphical models the parental dependency of a neuron is modelled by a non-linear (sigmoidal) function of the weighted parent states [3]. It turns out that the mean field approximation has the nice feature that it bounds the likelihood from below. This is useful for learning, since a maximisation of the bound either increases its accuracy or increases the likelihood for a pattern in the data set, which is the actual learning process.

In this article we show that it is possible to improve the mean field approximation

[†]http://www.mbfys.kun.nl/~bert

without losing the bounding properties. In section 2 we show the general theory to create a new bound using an existing one, which is applied to a Boltzmann machine in section 3. Boltzmann machines are another type of graphical models. In contrast with belief networks the connections are symmetric and not directed [4]. A mean field approximation for this type of neural networks was already described in [5]. An improvement of this approximation was found by Thouless, Anderson and Palmer in [6], which was applied to Boltzmann machines in [7]. Unfortunately, this so called TAP approximation is not a bound. We apply our method to the mean field approximation, which results in a third order bound. We prove that the latter is always tighter.

Due to the limited space it is not possible to discuss the third order bound for sigmoid belief networks in much detail. Instead, we show the general outline and focus more on the experimental results in section 5. Finally, in section 6, we present our conclusions.

## 2   Higher order bounds

Suppose we have a function $f_0(x)$ and a bound $b_0(x)$ such that $\forall_x \ f_0(x) \geq b_0(x)$. Let $f_1(x)$ and $b_1(x)$ be two primitive functions of $f_0(x)$ and $b_0(x)$

$$f_1(x) = \int \mathrm{d}x \ f_0(x) \quad \text{and} \quad b_1(x) = \int \mathrm{d}x \ b_0(x) \tag{1}$$

such that $f_1(\nu) = b_1(\nu)$ for some $\nu$. Note that we can always add an appropriate constant to the primitive functions such that they are indeed equal at $x = \nu$.

Since the surface under $f_0(x)$ at the left as well as at the right of $x = \nu$ is obviously greater than the surface under $b_0(x)$ and the primitive functions are equal at $x = \nu$ (by construction), we know

$$\begin{cases} f_1(x) \leq b_1(x) & \text{for } x \leq \nu \\ f_1(x) \geq b_1(x) & \text{for } x \geq \nu \end{cases} \tag{2}$$

or in shorthand notation $f_1(x) \lessgtr b_1(x)$. It is important to understand that even if $f_0(\nu) > b_0(\nu)$ the above result holds. Therefore we are completely free to choose $\nu$.

If we repeat this and let $f_2(x)$ and $b_2(x)$ be two primitive functions of $f_1(x)$ and $b_1(x)$, again such that $f_2(\nu) = b_2(\nu)$, one can easily verify that $\forall_x \ f_2(x) \geq b_2(x)$.

Thus given a lower bound of $f_0(x)$ we can create another lower bound. In case the given bound is a polynomial of degree $k$, the new bound is a polynomial of degree $k + 2$ with one additional variational parameter.

To illustrate this procedure, we derive a third order bound on the exponential function starting with the well known linear bound: the tangent of the exponential function at $x = \nu$. Using the procedure of the previous section we derive

$$\forall_{x,\nu} \ f_0(x) = \mathrm{e}^x \geq \mathrm{e}^\nu \left( 1 + x - \nu \right) = b_0(x) \tag{3}$$

$$f_1(x) = \mathrm{e}^x \lessgtr \mathrm{e}^\mu + \mathrm{e}^\nu \left( \left( 1 + \mu - \nu \right) \left( x - \mu \right) + \frac{1}{2} \left( x - \mu \right)^2 \right) = b_1(x) \tag{4}$$

$$\forall_{x,\mu,\lambda} \ f_2(x) = \mathrm{e}^x \geq \mathrm{e}^\mu \left\{ 1 + x - \mu + \mathrm{e}^\lambda \left( \frac{1 - \lambda}{2} \left( x - \mu \right)^2 + \frac{1}{6} \left( x - \mu \right)^3 \right) \right\} = b_2(x) \tag{5}$$

with $\lambda = \nu - \mu$.

## 3 Boltzmann machines

In this section we derive a third order lower bound on the partition function of a Boltzmann machine neural network using the results from the previous section. The probability to find a Boltzmann machine in a state $\vec{s} \in \{-1, +1\}^N$ is given by

$$P(\vec{s}) = \frac{1}{Z} \exp(-E(\vec{s})) = \frac{1}{Z} \exp\left(\frac{1}{2}\theta^{ij} s_i s_j + \theta^i s_i\right) \tag{6}$$

There is an implicit summation over all repeated indices (Einstein's convention). $Z = \sum_{\text{all } \vec{s}} \exp(-E(\vec{s}))$ is the normalisation constant known as the partition function which requires a sum over all, exponentially many states. Therefore this sum is intractable to compute even for rather small networks.

To compute the partition function approximately, we use the third order bound[1] from equation 5. We obtain

$$Z = \sum_{\text{all } \vec{s}} \exp(-E(\vec{s})) \geq \sum_{\text{all } \vec{s}} e^{\mu(\vec{s})} \left\{ 1 - \Delta E + e^{\lambda(\vec{s})} \left( \frac{1 - \lambda(\vec{s})}{2} \Delta E^2 - \frac{1}{6} \Delta E^3 \right) \right\} \tag{7}$$

where $\Delta E = \mu(\vec{s}) + E$. Note that the former constants $\mu$ and $\lambda$ are now functions of $\vec{s}$, since we may take different values for $\mu$ and $\lambda$ for each term in the sum. In principle these functions can take any form. If we take, for instance, $\mu(\vec{s}) = -E(\vec{s})$ the approximation is exact. This would lead, however, to the same intractability as before and therefore we must restrict our choice to those that make equation 7 tractable to compute. We choose $\mu(\vec{s})$ and $\lambda(\vec{s})$ to be linear with respect to the neuron states $s_i$:

$$\mu(\vec{s}) = \mu^i s_i + \mu^0 \quad \text{and} \quad \lambda(\vec{s}) = \lambda^i s_i + \lambda^0 \tag{8}$$

One may view $\mu(\vec{s})$ and $\lambda(\vec{s})$ as (the negative of) the energy functions for the Boltzmann distribution $P \sim \exp(\mu(\vec{s}))$ and $P \sim \exp(\lambda(\vec{s}))$. Therefore we will sometimes speak of 'the distribution $\mu(\vec{s})$'. Since these linear energy functions correspond to factorised distributions, we can compute the right hand side of equation 7 in a reasonable time, $\mathcal{O}(N^3)$.

To obtain the tightest bound, we may maximise equation 7 with respect to its variational parameters $\mu^0$, $\mu^i$, $\lambda^0$ and $\lambda^i$.

**A special case of the third order bound**

Although it is possible to choose $\lambda^i \neq 0$, we will set them to the suboptimal value $\lambda^i = 0$, since this simplifies equation 7 enormously. The reader should keep in mind, however, that all calculations could be done with non-zero $\lambda^i$. Given this choice we can compute the optimal values for $\mu^0$ and $\lambda^0$, given by

$$\begin{cases} \mu^0 = -\langle E + \mu^i s_i \rangle \\ \lambda^0 = -\dfrac{1}{3} \langle \Delta E^3 \rangle / \langle \Delta E^2 \rangle \end{cases} \tag{9}$$

where $\langle \cdot \rangle$ denotes an average over the (factorised) distribution $\mu(\vec{s})$. Using this solution the bound reduces to the simple form

$$\log Z \geq \log Z_\mu + \log \left\{ 1 + \frac{1}{2} e^{\lambda^0} \langle \Delta E^2 \rangle \right\} \tag{10}$$

where $Z_\mu$ is the partition function of the distribution $\mu(\vec{s})$. The term $\langle \Delta E^2 \rangle$ corresponds to the variance of $E + \mu^i s_i$ with respect to the distribution $\mu(\vec{s})$, since $\mu^0 = -\langle E + \mu^i s_i \rangle$. $\lambda^0$ is proportional to the third order moment according to (9). Explicit expressions for these moments can be derived with patience.

There is no explicit expression for the optimal $\mu^i$ as is the case with the standard mean field equations. An implicit expression, however, follows from setting the derivative with respect to $\mu^i$ to zero. We solved $\mu^i$ numerically by iteration. Wherever we speak of 'fully optimised', we refer to this solution for $\mu^i$.

**Connection with standard mean field and TAP**

We like to focus for a moment on the suboptimal case where $\mu^i$ correspond to the mean field solution, given by

$$\forall_i \quad m_i \overset{\text{def}}{=} \tanh \mu^i = \tanh\left(\theta^i + \theta^{ij} m_j\right) \tag{11}$$

For this choice for $\mu^i$ the $\log Z_\mu$ term in equation 10 is equal to the optimal mean field bound[2]. Since the last term in equation 10 is always positive, we conclude that the third order bound is always tighter than the mean field bound.

The relation between TAP and the third order bound is clear in the region of small weights. If we assume that $\mathcal{O}\left(\theta^{ij^3}\right)$ is negligible, a small weight expansion of equation 10 yields

$$\log Z \geq \log Z_\mu + \log\left\{1 + \frac{1}{2}e^{\lambda^0}\langle \Delta E^2 \rangle\right\} \approx \log Z_\mu + \frac{1}{4}\theta^{ij^2}\left(1 - m_i^2\right)\left(1 - m_j^2\right) \tag{12}$$

where the last term is equal to the TAP correction term [7]. Thus the third order bound tends to the TAP approximation for small weights. For larger weights, however, the TAP approximation overestimates the partition function, whereas the third order approximation is still a bound.

## 4 Sigmoid belief networks

In the previous section we saw how to derive a third order bound on the partition function. For sigmoid belief networks[3] we can use the same strategy to obtain a third order bound on the likelihood of the visible neurons of the network to be in a particular state. In this article, we present the rough outline of our method. The full derivation will be presented elsewhere.

It turns out that these graphical models are comparable to Boltzmann machines to a large extent. The energy function $E(\vec{s})$ (as in equation 6), however, differs for sigmoid belief networks:

$$-E(\vec{s}) = \theta^{ij} s_i s_j + \theta^i s_i - \sum_p \log 2\cosh\left(\theta^{pi} s_i + \theta^p\right) \tag{13}$$

The last term, known as the local normalisation, does not appear in the Boltzmann machine energy function. We have similar difficulties as with the Boltzmann machine, if we want to compute the log-likelihood given by

$$\log \mathcal{L} = \log \sum_{\vec{s} \in \text{Hidden}} P(\vec{s}) = \log \sum_{\vec{s} \in \text{Hidden}} \exp\left(-E(\vec{s})\right) \tag{14}$$

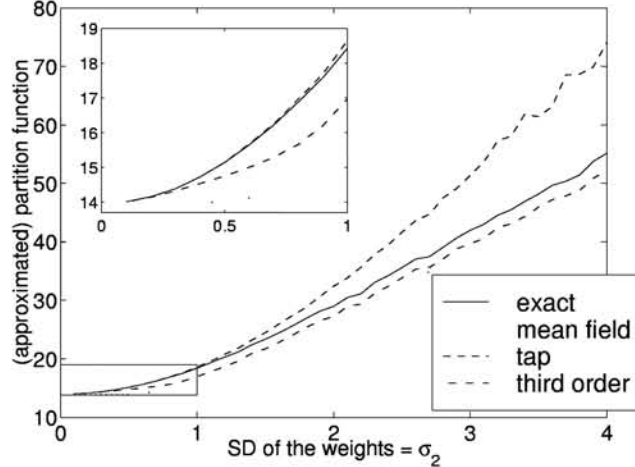

Figure 1: The exact partition function and three approximations: (1) Mean field, (2) TAP and (3) Fully optimised third order. The standard deviation of the thresholds is 0.1. Each point was averaged over a hundred randomly generated networks of 20 neurons. The inner plot shows the behaviour of the approximating functions for small weights.

In contrast with the Boltzmann machine, we are not finished by using equation 7 to bound $\mathcal{L}$. Due to the non-linear $\log 2\cosh$ term in the sigmoid belief energy, the so obtained bound is still intractable to compute. Therefore it is necessary to derive an additional bound such that the approximated likelihood is tractable to compute (this is comparable to the additional bound used in [2]). We make use of the concavity of the log function to find a straight line upper bound[4]: $\forall_{\xi} \ \log x \leq e^{\xi} x - \xi - 1$. We use this inequality to bound the $\log 2\cosh$ term in equation 13 for each $p$ separately, where we choose $\xi^p$ to be $\xi^p(\vec{s}) = \xi^{pi} s_i + \xi^p$. In this way we obtain a new energy function $\tilde{E}(\vec{s})$ which is an upper bound on the original energy. It is obvious that the following inequalities hold

$$\mathcal{L} = \sum_{\vec{s} \in \text{Hidden}} \exp\left(-E(\vec{s})\right) \geq \sum_{\vec{s} \in \text{Hidden}} \exp\left(-\tilde{E}(\vec{s})\right) \geq B\left(\tilde{E}, \mu, \lambda\right) \qquad (15)$$

where the last inequality is equal, apart from the tilde, to equation 7. It turns out that this bound has a worst case computational complexity of $\mathcal{O}(N^4)$, which makes it tractable for a large class of networks.

## 5    Results

### 5.1    Boltzmann machines

In this section we compare the third order bound for Boltzmann machines with (1) the exact partition function, (2) the standard mean field bound and (3) the TAP approximation. Therefore we created networks of $N = 20$ neurons with thresholds drawn from a Gaussian with zero mean and $\sigma_1 = 0.1$ and weights drawn from a Gaussian with zero mean and standard deviation $\sigma_2/\sqrt{N}$, a so called SK-model [8].

In figure 1 the exact partition function versus $\sigma_2$ is shown. In the same figure the mean field and fully optimised third order bound are shown together with the TAP approximation. For large $\sigma_2$ the exact partition function is linear in $\sigma_2$, whereas this is not necessarily the case for small $\sigma_2$ (see figure 1). In fact, in the absence of thresholds, the partition function is quadratic for small $\sigma_2$. Since TAP is based on a Taylor expansion in the weights upto second order, it is very accurate in the small weight region. However, as soon as the size of the weights exceeds the radius of convergence of this expansion (this occurs approximately at $\sigma_2 = 1$), the approximation diverges rapidly from the true value [9].

The mean field and third order approximation are both linear for large $\sigma_2$, which prevents that they cross the true partition function and would violate the bound. In fact, both approximations are quite close to the true partition function. For small weights ($\sigma_2 < 1$), however, we see that the third order bound is much closer to the exact curved form than mean field is.

## 5.2 Sigmoid belief networks

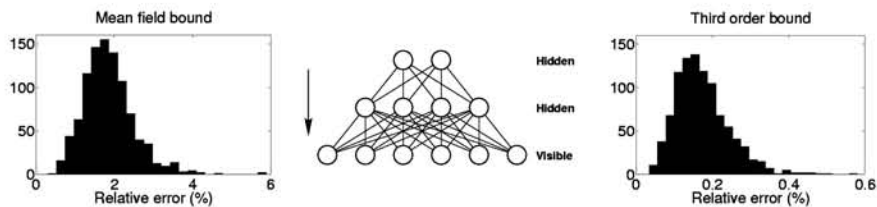

Figure 2: Histograms of the relative error for the toy network in the middle. The error of the third order bound is roughly ten times smaller than the error of the mean field bound.

Although a full optimisation of the variational parameters gives the tightest bound, it turns out that the computational complexity of this optimisation is quite large for sigmoid belief networks. Therefore, we use the mean field solution for $\mu^i$ (equation 11) instead. This can be justified since the most important error reduction is due to the use of the third order bound. From experimental results not shown here it is clear that a full optimisation has a share of only a few percent in the total gain.

To assess the error made by the various approaches, we use the same toy problem as in [2] and [10]. The network has a top layer of two neurons, a middle layer of four neurons and a lower layer of six visibles (figure 2). All neurons of two successive layers are connected with weights pointing downwards. Weights and thresholds are drawn from a uniform distribution over $[-1, 1]$.[5] We compute the likelihood when all visibles are clamped to $-1$. Since the network is rather small, we can compute the exact likelihood to compare the lower bound with.

In figure 2 a histogram of the relative error, $1 - \log B / \log \mathcal{L}$, is plotted for a thousand randomly generated networks. It is clear from the picture that for this toy problem the error is reduced by a factor ten. For larger weights, however, the effect is less, but still large enough to be valuable. For instance, if the weights are drawn from a uniform distribution over $[-2, 2]$, the error reduces by about a factor four on average and is always less than the mean field error.

# 6 Conclusions

We showed a procedure to find any odd order polynomial bound for the exponential function. A $2k-1$ order polynomial bound has $k$ variational parameters. For the third order bound these are $\mu$ and $\lambda$. We can use this result to derive a bound on the partition function, where the variational parameters can be seen as energy functions for probability distributions. If we choose those distributions to be factorised, we have $(N+1)k$ new variational parameters. Since the approximating function is a bound, we may maximise it with respect to all these parameters.

In this article we restricted ourselves to the third order bound, although an extension to any odd order bound is possible. Third order is the next higher order bound to naive mean field. We showed that this bound is strictly better than the mean field bound and tends to the TAP approximation for small weights. For larger weights, however, the TAP approximation crosses the partition function and violates the bounding properties.

We saw that the third order bound gives an enormous improvement compared to mean field. Our results are comparable to those obtained by the structured approach in [10]. The choice between third order and variational structures, however, is not exclusive. We expect that a combination of both methods is a promising research direction to obtain the tightest tractable bound.

### Acknowledgements

This research is supported by the Technology Foundation STW, applied science devision of NWO and the technology programme of the Ministry of Economic Affairs.

## Footnotes

\*http://www.mbfys.kun.nl/~martijn

[1] Using the first order bound from equation 3 results in the standard mean field bound.

[2]Be aware of the fact that $\mu(\vec{s})$ contains the parameter $\mu^0 = -\langle E + \mu^i s_i \rangle$. This gives an important contribution to the expression for $\log Z_\mu$.

[3]A detailed description of these networks can be found in [3].

[4]This bound is also derivable using the method from section 2 with $f_0(x) = \frac{1}{x^2} \geq 0$.

[5]The original toy problem in [2] used a 0/1-coding for the neuron activity. To be able to compare the results, we transform the weights and thresholds to the $-1/+1$-coding used in this article.

# References

[1] J. Pearl. *Probabilistic Reasoning in Intelligent Systems*, chapter 8.2.1, pages 387–390. Morgan Kaufmann, San Francisco, 1988.

[2] S.K. Saul, T.S. Jaakkola, and M.I. Jordan. Mean field theory for sigmoid belief networks. Technical Report 1, Computational Cognitive Science, 1995.

[3] R. Neil. Connectionist learning of belief networks. *Artificial intelligence*, 56:71–113, 1992.

[4] D. Ackley, G. Hinton, and T. Sejnowski. A learning algorithm for Boltzmann machines. *Cognitive Science*, 9:147–169, 1985.

[5] C. Peterson and J. Anderson. A mean field theory learning algorithm for neural networks. *Complex systems*, 1:995–1019, 1987.

[6] D.J. Thouless, P.W. Andersson, and R.G. Palmer. Solution of 'solvable model of a spin glass'. *Philisophical Magazine*, 35(3):593–601, 1977.

[7] H.J. Kappen and F.B. Rodríguez. Boltzmann machine learning using mean field theory and linear response correction. In M.S. Kearns, S.A. Solla, and D.A. Cohn, editors, *Advances in Neural Information Processing Systems*, volume 11, pages 280–286. MIT Press, 1999.

[8] D. Sherrington and S. Kirkpatrick. Solvable model of a spin-glass. *Physical Review Letters*, 35(26):1793–1796, 12 1975.

[9] M.A.R. Leisink and H.J. Kappen. Validity of TAP equations in neural networks. In ICANN 99, volume 1, pages 425–430, ISBN 0 85296 721 7, 1999. Institution of Electrical Engineers, London.

[10] D. Barber and W. Wiegerinck. Tractable variational structures for approximating graphical models. In M.S. Kearns, S.A. Solla, and D.A. Cohn, editors, *Advances in Neural Information Processing Systems*, volume 11, pages 183–189. MIT Press, 1999.
